# Fragment completion in humans and machines

**David Jacobs**
NEC Research Institute
4 Independence Way, Princeton, NJ 08540
*dwj@research.nj.nec.com*

**Bas Rokers**
Psychology Department at UCLA
PO Box 951563, Los Angeles, CA 90095
*rokers@psych.ucla.edu*

**Archisman Rudra**
CS Department at NYU
251 Mercer St., New York, NY 10012
*archi@cs.nyu.edu*

**Zili Liu**
Psychology Department at UCLA
PO Box 951563, Los Angeles CA 90095
*zili@psych.ucla.edu*

## Abstract

Partial information can trigger a complete memory. At the same time, human memory is not perfect. A cue can contain enough information to specify an item in memory, but fail to trigger that item. In the context of word memory, we present experiments that demonstrate some basic patterns in human memory errors. We use cues that consist of word fragments. We show that short and long cues are completed more accurately than medium length ones and study some of the factors that lead to this behavior. We then present a novel computational model that shows some of the flexibility and patterns of errors that occur in human memory. This model iterates between bottom-up and top-down computations. These are tied together using a Markov model of words that allows memory to be accessed with a simple feature set, and enables a bottom-up process to compute a probability distribution of possible completions of word fragments, in a manner similar to models of visual perceptual completion.

## 1 Introduction

This paper addresses the problem of retrieving items in memory from partial information. Human memory is remarkable for its flexibility in handling a wide range of possible retrieval cues. It is also very accurate, but not perfect; some cues are more easily used than others. We hypothesize that memory errors occur in part because a trade-off exists between memory accuracy and the complexity of neural hardware needed to perform complicated memory tasks. If this is true, we can gain insight into mechanisms of human memory by studying the patterns of errors humans make, and we can model human memory with systems that produce similar patterns as a result of constraints on computational resources.

We experiment with word memory questions of the sort that arise in a game called *superghost*. Subjects are presented with questions of a form: '*p*l*c*'. They must find a valid English word that matches this query, by replacing each '*' with zero or more letters. So for this example, 'place', 'application', and 'palace' would all be valid answers. In ef-

fect, the subject is given a set of letters and must think of a word that contains all of those letters, in that order, with other letters added as needed.

Most of the psychological literature on word completion involves the effects of priming certain responses with recent experience (Shacter and Tulving[18]). However, priming is only able to account for about five percent of the variance in a typical fragment completion task (Olofsson and Nyberg[13], Hintzman and Hartry[6]). We describe experiments that show that the difficulty of a query depends on what we call its *redundancy*. This measures the extent to which all the letters in the query are needed to find a valid answer. We show that when we control for the redundancy of queries, we find that the difficulty of answering questions increases with their length; queries with many letters tend to be easy only because they tend to be highly redundant. We then describe a model that mimics these and other properties of human memory.

Our model is based on the idea that a large memory system can gain efficiency by keeping the comparison between input and items in memory as simple as possible. All comparisons use a small, fixed set of features. To flexibly handle a range of queries, we add a bottom-up process that computes the probability that each feature is present in the answer, given the input and a generic, Markov model of words. So the complexity of the bottom-up computation does not grow with the number of items in memory. Finally, the system is allowed to iterate between this bottom up and a top down process, so that a new generic model of words is constructed based on a current probability distribution over all words in memory, and this new model is combined with the input to update the probability that each feature is present in the answer.

Previous psychological research has compared performance of word-stem and word-fragment completion. In the former a number of letters (i.e. a fragment) is given beginning with the first letter(s) of the word. In the latter, the string of letters given may begin at any point in the word, and adjacent letters in the fragment do not need, but may, be adjacent in the completed word. For example, for stem completion the fragment "str" may be completed into "string", but for fragment completion also into "satire". Performance for word-fragment completion is lower than word-stem completion (Olofsson and Nyberg[12]). In addition words, for which the ending fragment is given, show performance closer to word-stem completion than to word-fragment completion (Olofsson and Nyberg[13]).

Seidenberg[17] proposed a model based on tri-grams. Srinivas et al.[21] indicate that assuming orthographic encoding is in most cases sufficient to describe word completion performance in humans. Orthographic Markov models of words have often been used computationally, as, for example, in Shannon's[19] famous work. Following this work, our model is also orthographic. We find that a bigram rather than a trigram representation is sufficient, and leads to a simpler model.

Contradicting evidence exists for the influence of fragment length on word completion. Oloffsson and Nyberg [12] failed to find a difference between two and three letter fragments on words of length of five to eight letters. However this might have been due to the fact that in their task, each fragment has a unique completion.

Many recurrent neural networks have been proposed as models of associative memory (Anderson[1] contains a review). Perhaps most relevant to our work are models that use an input query to activate items from a complete dictionary in memory, and then use these items to alter the activations of the input. For example, in the Interactive Activation model of Rumelhart and McClelland[16], the presence of letters activates words, which boost the activity of the letters they contain. In Adaptive Resonance models (Carpenter and Grossberg[3]) activated memory items are compared to the input query and de-activated if they do not match. Also similar in spirit to our approach is the bidirectional model of Kosko[10] (for more recent work see, eg., Sommer and Palm[20]). Other models iteratively combine top-down and bottom-up information (eg., Hinton et al.[5], Rao and Ballard[14]),

although these are not used as part of a memory system with complete items stored in memory.

Our model differs from all of these in using a Markov model as an intermediate layer between the input and the dictionary. This allows the model to answer superghost queries, and leads to different computational mechanisms that we will detail. We find that superghost queries seem more natural to people than associative memory word problems (compare the superghost query "think of a word with an $a$" to the associative memory query "think of a word whose seventh letter is an $a$"). However, it is not clear how to extend most models of associative memory to handle superghost problems.

Our use of features is more related to feedforward neural nets, and especially the "information bottleneck" approach of Tishby, Pereira and Bialek[22] (see also Baum, et al.[2]). Our work differs from feedforward methods in that our method is iterative, and uses features symmetrically to relate the memory to input in both directions.

Our approach is also related to work on visual object recognition that combines perceptual organization and top-down knowledge (see Ullman[23]). Our model is inspired by Mumford's[11] and Williams and Jacobs'[24] use of Markov models of contours for bottom-up perceptual completion.

Especially relevant to our work is that of Grimes and Mozer[4]. Simultaneous with our work ([8]) they use a bigram model to solve anagram problems, in which letters are unscrambled to match words in a dictionary. They also use a Markov model to find letter orderings that conform with the statistics of English spelling. Their model is quite different in how this is done, due to the different nature of the anagram problem. They view anagram solving as a mix of low-level processing and higher level cognitive processes, while it is our goal to focus just on lower level memory.

## 2  Experiments with Human Subjects

In our experiments, fragments and matching words were drawn from a large standard corpus of English text. The *frequency* of a word is the number of times it appears in this corpus. The frequency of a fragment is the sum of the frequency of all words that the fragment matches. We used fragments of length two to eight, discarding any fragments with frequency lower than one thousand.

Fragments selected for an experiment were presented in random order. In our first experiment we systematically varied the length of the fragments, but otherwise selected them from a uniform, random distribution. Consequently, shorter fragments tended to match more words, with greater total frequency. In the second experiment, fragments were selected so that a uniform distribution of frequencies was ensured over all fragment lengths. For example, we used length two fragments that matched unusually few words. As a result the average frequency in experiment two is also much lower than in experiment one.

A fragment was presented on a computer screen with spaces interspersed, indicating the possibility of letter insertion. The subject was required to enter a word that would fit the fragment. A subject was given 10 seconds to produce a completion, with the possibility to give up. For each session 50 fragments were presented, with a similar number of fragments of each length.

Reaction times were recorded by measuring the time elapsed between the fragment first appearing on screen and the subject typing the first character of a matching word. Words that did not match the fragment or did not exist in the corpus were marked as not completed.

Each experiment was completed by thirty-one subjects. The subjects were undergraduate students at Rutgers University, participating in the experiment for partial credit. Total time

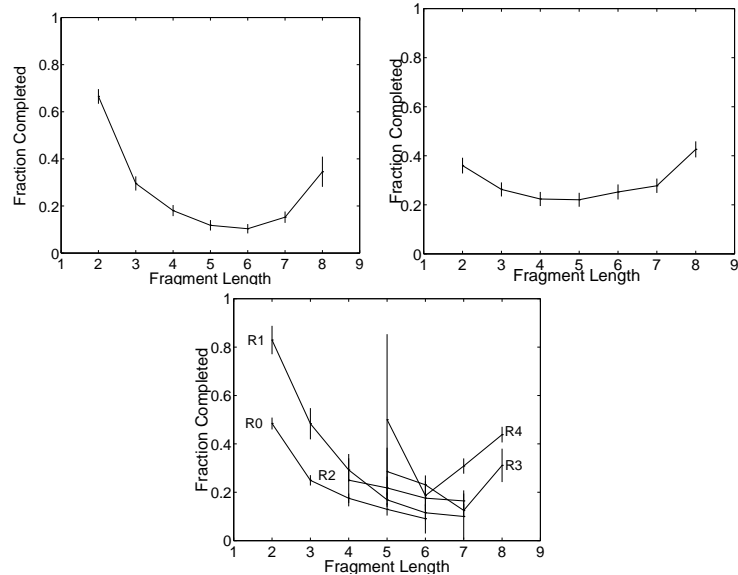

Figure 1: Fragment completion as a function of fragment length for randomly chosen cues (top-left) and cues of equal frequency (top-right). On the bottom, the equal frequency cues are divided into five groups, from least redundancy (R0) to most (R5) .

spent on the task varied from 15 minutes to close to one hour.

## Results

For each graph we plot the number of fragments completed divided by the number of fragments presented (Figure 1). Error bars are calculated as $\sqrt{(p - p^2)/n}$, where $p$ is the percent correct in the sample, and $n$ is the number of trials. This assumes that all decisions are independent and correct with probability $p$; more precise results can be obtained by accounting for between-subject variance, but roughly the same results hold.

For random, uniformly chosen fragments, there is a U-shaped dependence of performance on length. Controlling for frequency reduces performance because on average lower frequency fragments are selected. The U-shaped curve is flattened, but persists; hence U-shaped performance is not just due to frequency

Finally, we divide the fragments from the two experiments into five groups, according to their *redundancy*. This is a rough measure of how important each letter is in finding a correct answer to the overall question. It is the probability that if we randomly delete a letter from the fragment and find a matching word, that this word will match the full fragment. Specifically, let $f$ denote the frequency of a query fragment of length $k$ (total frequency of words that match it). Let $f_i$ denote the frequency of the fragment that results when we delete the $i$'th letter from the query (note, $f_i \geq f$). Then redundancy is: $k * f / (f_1 + ... + f_k)$. In all cases where there is a significant difference, greater redundancy leads to better performance. In almost all cases, when we control for redundancy performance decreases with length. We will discuss the implications of these experiments after describing corresponding experiments with our model.

# 3 Using Markov Models for Word Retrieval

We now describe a model of word memory in which matching between the query and memory is mediated by a simple set of features. Specifically, we use bigrams (adjacent pairs of letters) as our feature set. We denote the beginning and end of a word using the symbols '0' and '1', respectively, so that bigram probabilities also indicate how often individual letters begin or end a word. Bottom up processing of a cue is done using this as a Markov model of words. Then bigram probabilities are used to trigger words in memory that might match the query.

Our algorithm consists of three steps. First, we compute a prior distribution on how likely each word in memory is to match our query. In our simulations, we just use a uniform distribution. However, this distribution could reflect the frequency with which each word occurs in English. It could also be used to capture priming phenomena; for example, if a word has been recently seen, its prior probability could increase, making it more likely that the model would retrieve this word. Then, using these we compute a probability that each bigram will appear if we randomly select a bigram from a word selected according to our prior distribution.

Second, we use these bigram probabilities as a Markov model, and compute the expected number of times each bigram will occur in the answer, conditioned on the query. That is, as a generic model of words we assume that each letter in the word depends on the adjacent letters, but is conditionally independent of all others. This conditional independence allows us to decompose our problem into a set of small, independent problems. For example, consider the query '*p*l*c*'. Implicitly, each query begins with '0' and ends with '1', so the expected number of times any bigram will appear in the completed word is the sum of the number of times it appears in the completions of the fragments: '0*p', 'p*l', 'l*c', and 'c*1'.

To compute this, we assume a prior distribution on the number of letters that will replace a '*' in the completed word. We use an exponential model, setting the probability of $n$ letters to be $\frac{1}{2}^{n+1}$ (in practice we truncate $n$ at 5 and normalize the probabilities). A similar model is used in the perceptual completion of contours ([11, 24]). Using these priors, it becomes straightforward to compute a probability distribution on the bigrams that will appear in the completed cue. For a fixed $n$, we structure this problem as a belief net with $n + 1$ bigrams, and each bigram depending on only its neighbors. The conditional probability of each bigram given its neighbor comes from the Markov model, and we can solve the problem with belief propagation.

Beginning the third step of the algorithm, we know the expected number of times that each bigram appears in the completed cue. Each bigram then votes for all words containing that bigram. The weight of this vote is the expected number of times each bigram appears in the completed cue, divided by the prior probability of each bigram, computed in step 1. We combine these votes multiplicatively. We update the prior for each word as the product of these votes with the previous probability. We can view this an approximate computation of the probability of each word being the correct answer, based on the likelihood that a bigram appears in the completed cue, and our prior on each word being correct.

After the third step, we once again have a probability that each word is correct, and can iterate, using this probability to initialize step one. After a small number of iterations, we terminate the algorithm and select the most probable word as our answer. Empirically, we find that the answer the algorithm produces often changes in the first one or two iterations, and then generally remains the same. The answer may or may not actually match the input cue, and by this we judge whether it is correct or incorrect.

We can view this algorithm as an approximate computation of the probability that each

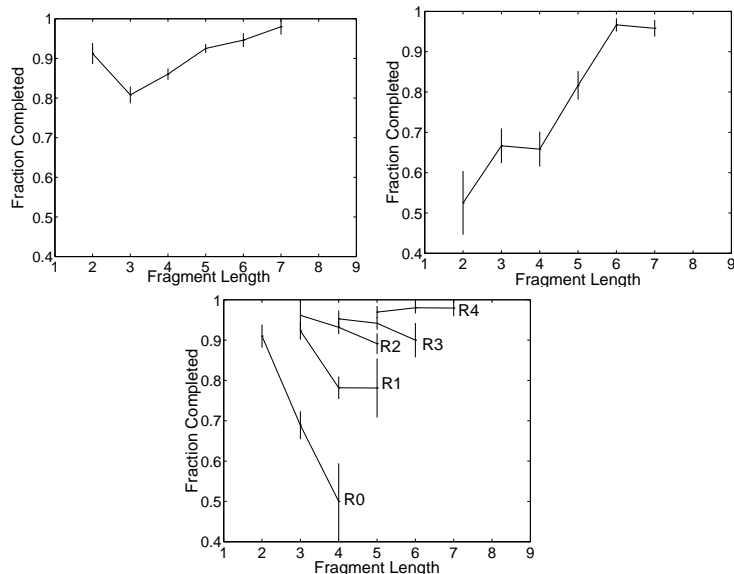

Figure 2: Performance as a function of cue length, for cues of frequency between 4 and 22 (top-left) and between 1 and 3 (top-right). On the bottom, we divide the first set of cues into five groups ranging from the least redundant (R0) to the most (R4).

word matches the cue, where the main approximation comes from using a small set of features to bring the cue into contact with items in memory. Denote the number of features by $F$ (with a bigram representation, $F = 728$), the number of features in each word by $m$ (ie., the word length plus one), the number of words by $w$, and the maximum number of blanks replacing a '*' by $n$. Then steps one and three require O(mw) computation, and step two requires O(Fn) computation. In a neural network, the primary requirement would be bidirectional connections between each feature (bigram) and each item in memory. Therefore, computational simplicity is gained by using a small feature set, at the cost of some approximation in the computation.

**Experiments**

We have run experiments to compare the performance of this model to that of human subjects. For simplicity, we used a memory of 6,040 words, each with eight characters. First, we simulated the conditions described in Olofsson and Nyberg[12] comparing word stem and word fragment completion. To match their experiments, we used a modified algorithm that handled cues in which the number of missing letters can be specified. We used cues that specified the first three letters of a word, the last three letters, or three letters scattered throughout the word. The algorithm achieved accuracy of 95% in the first case, 87% in the second, and 80% in the third. This qualitatively matches the results for human subjects. Note that our algorithm treats the beginning and end of words symmetrically. Therefore, the fact that it performs better when the first letters of the word are given than when the last are given is due to regularities in English spelling, and is not built into the algorithm.

Next we simulated conditions comparable to our own experiments on human subjects, using superghost cues. First we selected cues of varying length that match between four and twenty-two words in the dictionary. Figure 2-top-left shows the percentage of queries the algorithm correctly answered, for cues of lengths two to seven. This figure shows a U-shaped performance curve qualitatively similar to that displayed by human subjects.

We also ran these experiments using cues that matched one to three words (Figure 2-top-right). These very low frequency cues did not display this U-shaped behavior. The algorithm performs differently on fragments with very low frequency because in our corpus the shorter of these cues had especially low redundancy and the longer fragments had especially high redundancy, in comparison to fragments with frequencies between 4 and 22. Next (Figure 2-bottom) we divided the cues into five groups of equal size, according to their redundancy. We can see that performance increases with redundancy and decreases with cue length.

## Discussion

Our experiments indicate two main effects in human word memory that our model also shares. First, performance improves with the redundancy of cues. Second, when we control for this, performance drops with cue length. Since redundancy tends to increase with cue length, this creates two conflicting tendencies that result in a U-shaped memory curve. We conjecture that these factors may be present in many memory tasks, leading to U-shaped memory curves in a number of domains.

In our model, the fact that performance drops with cue length is a result of our use of a simple feature set to mediate matching the cue to words in memory. This means that not all the information present in the cue is conveyed to items in memory. When the length of a cue increases, but its redundancy remains low, all the information in the cue remains important in getting a correct answer, but the amount of information in the cue increases, making it harder to capture it all with a limited feature set. This can account for the performance of our model; similar mechanisms may account for human performance as well. On the other hand, the extent to which redundancy grows with cue length is really a product of the specific words in memory and the cues chosen. Therefore, the exact shape of the performance curve will also depend on these factors. This may partly explain some of the quantitative differences between our model and human performance.

Finally, we also point out that our measure of redundancy is rather crude. In particular, it tends to saturate at very high or very low levels. So, for example, if we add a letter to a cue that is already highly redundant, the new letter may not be needed to find a correct answer, but that is not reflected by much of an increase in the cue's redundancy.

## 4   Conclusions

We have proposed superghost queries as a domain for experimenting with word memory, because it seems a natural task to people, and requires models that can flexibly handle somewhat complicated questions. We have shown that in human subjects, performance on superghost improves with the redundancy of a query, and otherwise tends to decrease with word length. Together, these effects results in a U-shaped performance curve.

We have proposed a computational model that uses a simple, generic model of words to map a superghost query onto a simple feature set of bigrams. This means that somewhat complicated questions can be answered while keeping comparisons between the fragments and words in memory very simple. Our model displays the two main trends we have found in human memory. It also does better at word stem completion than word fragment completion, which agrees with previous work on human memory. Future work will investigate the modification of our model to account for priming effects in memory.

## References

[1]  J. Anderson. *An Introduction to Neural Networks*, MIT Press, Cambridge MA. 1995.

[2] E. Baum, J. Moody and F. Wilczek. "Internal Representations for Associative Memory," *Biological Cybernetics*, 59:217-228, 1988.

[3] G. Carpenter, and S. Grossberg. "ART 2: Self-Organization of Stable Category Recognition Codes for Analog Input Patterns," *Applied Optics*, 26:4919-4930, 1987.

[4] D. Grimes and M. Mozer. "The interplay of symbolic and subsymbolic processes in anagram problem solving," *NIPS*, 2001.

[5] G. Hinton, P. Dayan, B. Frey, and R. Neal. "The 'Wake-Sleep' Algorithm for Unsupervised Neural Networks," *Science*, 268:1158-1161, 1995.

[6] D.L. Hintzman and A.L. Hartry. Item effects in recognition and fragment completion: Contingency relations vary for different sets of words. *JEP: Learning, Memory and Cognition*, 17: 341-345, 1990.

[7] J. Hopfield. "Neural networks and Physical Systems with Emergent Collective Computational Abilities." *Proc. of the Nat. Acad. of Science*, 79:2554-2558, 1982.

[8] D. Jacobs and A. Rudra. "An Iterative Projection Model of Memory," NEC Research Institute Technical Report, 2000.

[9] G.V. Jones. Fragment and schema models for recall. *Memory and Cognition*, 12(3):250-63, 1984.

[10] B. Kosko. "Adaptive Bidirectional Associative Memory", *Applied Optics*, 26(23):4947-60, 1987.

[11] D. Mumford. "Elastica and Computer Vision." C. Bajaj (Ed), *Algebraic Geometry and its Applications* New York: Springer-Verlag. 1994.

[12] U. Olofsson and L. Nyberg. Swedish norms for completion of word stems and unique word fragments. *Scandinavian Journal of Psychology*, 33(2):108-16, 1992.

[13] U. Olofsson and L. Nyberg. Determinants of word fragment completion. *Scandinavian Journal of Psychology*, 36(1):59-64, 1995.

[14] R. Rao and D. Ballard. "Dynamic Model of Visual Recognition Predicts Neural Response Properties in the Visual Cortex," *Neural Computation*, 9(4):721-763, 1997.

[15] R.H. Ross and G.H. Bower. Comparisons of models of associative recall. *Memory and Cognition*, 9(1):1-16, 1981.

[16] D. Rumelhart and J. McClelland. "An interactive activation model of context effects in letter perception: part 2. The contextual enhancement effect and some tests and extensions of the model", *Psychological Review*, 89:60-94, 1982.

[17] M.S. Seidenberg. Sublexical structures in visual word recognition: Access units or orthographic redundancy? In M. Coltheart (Ed.), *Attention and performance XII*, 245-263. Hillsdale, NJ: Erlbaum. 1987.

[18] D.L. Shacter and E. Tulving. *Memory systems.* Cambridge, MA: MIT Press. 1994.

[19] C. Shannon. "Prediction and Entropy of Printed English," *Bell Systems Technical Journal*, 30:50-64, 1951.

[20] Sommer, F., and Palm, G., 1997, *NIPS*:676-681.

[21] K. Srinivas, H.L. Roediger 3d and S. Rajaram. The role of syllabic and orthographic properties of letter cues in solving word fragments. *Memory and Cognition*, 20(3):219-30, 1992.

[22] N. Tishby, F. Pereira and W. Bialek. "The Information Bottleneck Method," 37th Allerton Conference on Communication, Control, and Computing. 1999.

[23] S. Ullman. *High-level Vision*, MIT Press, Cambridge, MA. 1996.

[24] L. Williams & D. Jacobs. "Stochastic Completion Fields: A Neural Model of Illusory Contour Shape and Salience". *Neural Computation*, 9:837–858, 1997.

## Acknowledgements

The authors would like to thank Nancy Johal for her assistance in conducting the psychological experiments presented in this paper.
